# fMRI-Based Inter-Subject Cortical Alignment Using Functional Connectivity

**Bryan R. Conroy[1]    Benjamin D. Singer[2]    James V. Haxby[3]***    **Peter J. Ramadge[1]**
[1] Department of Electrical Engineering, [2] Neuroscience Institute, Princeton University
[3] Department of Psychology, Dartmouth College

## Abstract

The inter-subject alignment of functional MRI (fMRI) data is important for improving the statistical power of fMRI group analyses. In contrast to existing anatomically-based methods, we propose a novel multi-subject algorithm that derives a functional correspondence by aligning spatial patterns of functional connectivity across a set of subjects. We test our method on fMRI data collected during a movie viewing experiment. By cross-validating the results of our algorithm, we show that the correspondence successfully generalizes to a secondary movie dataset not used to derive the alignment.

## 1  Introduction

Functional MRI (fMRI) studies of human neuroanatomical organization commonly analyze fMRI data across a population of subjects. The effective use of this data requires deriving a spatial correspondence across the set of subjects, i.e., the data must be aligned, or registered, into a common coordinate space. Current inter-subject registration techniques derive this correspondence by aligning anatomically-defined features, e.g. major sulci and gyri, across subjects, either in the volume or on extracted cortical surfaces. Talairach normalization [1], for example, derives a piecewise affine transformation by matching a set of major anatomical landmarks in the brain volume. More advanced techniques match a denser set of anatomical features, such as cortical curvature [2], and derive nonlinear transformations between a reference space and each subject's cortical surface.

It is known, however, that an accurate inter-subject *functional correspondence* cannot be derived using only anatomical features, since the size, shape and anatomical location of functional loci vary across subjects [3], [4]. Because of this deficiency in current alignment methods, it is common practice to spatially smooth each subject's functional data prior to a population based analysis. However, this incurs the penalty of blurring the functional data within and across distinct cortical regions. Thus, the functional alignment of multi-subject fMRI data remains an important problem.

We propose to register functional loci directly by using anatomical and functional data to learn an inter-subject cortical correspondence. This approach was first explored in [5], where subject cortices were registered by maximizing the inter-subject correlation of the functional response elicited by a common stimulus (a movie viewing). In essence, the correspondence was selected to maximize the correlation of the fMRI time series between subjects. This relies on the functional response being time-locked with the experimental stimulus. Large regions of visual and auditory cortex stimulated by a movie viewing do indeed show consistent inter-subject synchrony [6]. However, other areas in the intrinsic [7] or default [8] system fail to exhibit significant correlations across repeated stimulus trials. The technique of [5] is hence not expected to improve alignment in these intrinsic regions.

In contrast to [5], we propose to achieve inter-subject alignment by aligning intra-subject patterns of cortical *functional connectivity*. By functional connectivity, we mean within-subject similarity of

the temporal response of remote regions of cortex [9]. This can be estimated from fMRI data, for example, by correlating the functional time series between pairs of cortical nodes within a subject. This yields a dense set of functional features for each subject from which we learn an inter-subject correspondence. Unlike other functional connectivity work (see e.g. [10]), we define connectivity between pairs of cortical nodes rather than with respect to anatomical regions of interest. Our approach is inspired by studies showing that the patterns of functional connectivity in the intrinsic network are consistent across subjects [7], [11]. This suggests that our method has the potential to learn an inter-subject functional correspondence within both extrinsic and intrinsic cortical networks.

In summary, we formulate a multi-subject cortical alignment algorithm that minimizes the difference between functional connectivity vectors of corresponding cortical nodes across subjects. We do so by learning a dense-deformation field on the cortex of each subject, suitably regularized to preserve cortical topology [2]. Our key contributions are: a) the novel alignment objective, b) a principled algorithm for accomplishing the alignment, and c) experimental verification on fMRI data.

The paper is organized as follows. In §2 we formulate the multi-subject alignment problem, followed by a detailed exposition of the algorithm in §3 and §4. Finally, we exhibit results of the algorithm applied to multi-subject fMRI data in §5 and draw conclusions in §6.

## 2 Formulation of the Multi-Subject Alignment Problem

For each subject we are given volumetric anatomical MRI data and fMRI data. The anatomical data is used to extract a two-dimensional surface model of cortex. This greatly facilitates cortical based analysis and subsequent visualization [12], [13], [14]. Cortex is segmented, then each cortical hemisphere is inflated to obtain a smooth surface, which is projected to the sphere, $S^2$, represented by a discrete spherical mesh $M_s = \{p_k \in S^2; \ 1 \leq k \leq N_v/2\}$. The two cortical hemispheres are hence modeled by the disjoint union $S = S^2 \uplus S^2$, represented by the corresponding disjoint union of mesh points $M = M_s \uplus M_s$. Anatomical cortical features, such as cortical curvature, are functions $D_a \colon S \to \mathbf{R}^{N_a}$ sampled on $M$. Thus, our analysis is restricted to cortex only.

The fMRI volumeric data is first aligned with the anatomical scan, then mapped onto $S$. This assigns each mesh node $p_k \in M$ a "volumetric cortical voxel" $v_k \in \mathbb{R}^3$, with associated functional time series $f_k \in \mathbb{R}^{N_t}$. The functional time series data is then a function $D_f \colon S \to \mathbf{R}^{N_t}$ sampled on $M$.

As indicated in the introduction, we do not directly register the fMRI time series but instead register the functional connectivity derived from the time series. Let $\sigma(f_1, f_2)$ denote a similarity measure on pairs of time series $f_1, f_2 \in \mathbf{R}^{N_t}$. A useful example is empirical correlation: $\sigma(f_1, f_2) = \mathrm{corr}(f_1, f_2)$; another possibility is an estimate of the mutual information between the pairwise entries of $f_1, f_2$. Define the *functional connectivity* of the fMRI data under $\sigma$ as the map $C(p_i, p_j) = \sigma(D_f(p_i), D_f(p_j))$, i.e., the similarity of the functional times series at the pairs of cortical nodes. Functional connections both within and across cortical hemispheres are considered. Functional connectivity can be conceptualized as the adjacency matrix of an edge-weighted graph on all cortical nodes. The edge between nodes $p_i, p_j$ is weighted by the pairwise similarity measure $\sigma(f_i, f_j)$ codifying the *functional similarity* of $p_i$ and $p_j$. In the case of correlation, $C$ is the correlation matrix of the time series data. For typical values of $N_v$ ($\approx 72,000$), the functional connectivity data structure is huge. Hence we need efficient mechanisms for working with $C$.

We are given the data discussed above for $N_s$ subjects. Subject $k$'s training data is specified by samples of the functions $D_{a,k} \colon S_k \to \mathbf{R}^{N_a}$, $D_{f,k} \colon S_j \to \mathbf{R}^{N_t}$, and the derived functional connectivity $C_k$, all sampled on the mesh $M_k$, $k = 1, \ldots, N_s$. Our objective is to learn a relation consisting of $N_s$-tuples of corresponding points across the set of cortices. To do so, we could select a node from $M_1$ for subject 1 and learn the corresponding points on the cortices of the remaining $N_s - 1$ subjects through smooth and invertible mappings $g_k \colon S_1 \to S_k, k = 2, \ldots, N_s$. However, this arbitrarily and undesirably gives special status to one subject. Instead, we introduce a reference model $S_{\mathrm{ref}} = S^2 \uplus S^2$ with mesh $M_{\mathrm{ref}}$. For each node $p \in M_{\mathrm{ref}}$ on $S_{\mathrm{ref}}$, we seek to learn the $N_s$-tuple of corresponding points $(g_1(p), g_2(p), \ldots, g_{N_s}(p))$, parameterized by $g_k \colon S_{\mathrm{ref}} \to S_k, k = 1, \ldots, N_s$.

In general terms, we can now summarize our task as follows: use the functional connectivity data $C_k$, in conjunction with the anatomical data $D_{a,k}$, $k = 1, \ldots, N_s$, to estimate warping functions $\{g_k \colon k = 1, \ldots, N_s\}$, subject to specified regularity conditions, that bring some specified balance of anatomy and functional connectivity into alignment across subjects. That said, for the remainder

of the paper we restrict attention to aligning only functional connectivity across subjects. There is no doubt that anatomy must be an integral part of a full solution; but that aspect is not new, and is already well understood. Restricting attention to the alignment of functional connectivity will allow us to concentrate on the most novel and important aspects of our approach.

To proceed, assume a reference connectivity $C_{\text{ref}}$, such that for each subject $k = 1, \ldots, N_s$,

$$C_k(g_k(p_i), g_k(p_j)) = C_{\text{ref}}(p_i, p_j) + \epsilon_k(p_i, p_j), \quad p_i, p_j \in M_{\text{ref}} \tag{1}$$

where $C_k(g_k(p_i), g_k(p_j)) = \sigma(D_{f,k}(g_k(p_i)), D_{f,k}(g_k(p_j)))$, and $\epsilon_k$ is zero-mean random noise. Since $g_k(p)$ may not be a mesh point, computation of $D_{f,k}(g_k(p))$ requires interpolation of the time series using mesh nodes in a neighborhood of $g_k(p)$. This will be important as we proceed.

Given (1), we estimate $g$ by maximizing a regularized log likelihood:

$$\hat{g} = \arg \max_{g=(g_1, \cdots, g_{N_s})} \log P(C_1, \ldots, C_{N_s}|g) - \lambda \sum_k \text{Reg}(g_k) \tag{2}$$

where $\text{Reg}(g_k)$ constrains each warping function $g_k$ to be smooth and invertible. Here, we will focus on the log likelihood term and delay the discussion of regularization to §3. Optimization of (2) is complicated by the fact that $C_{\text{ref}}$ is a latent variable, so it must be estimated along with $g$. We use Expectation-Maximization to iteratively alternate between computing an expectation of $C_{\text{ref}}$ (E-step), and a maximum likelihood estimate of $g$ given both the observed and estimated unobserved data (M-step) [15]. In the E-step, the expectation of $C_{\text{ref}}$, $\overline{C}_{\text{ref}}$, conditioned on the current estimate of $g$, $\hat{g}$, is computed by averaging the connectivity across subjects:

$$\overline{C}_{\text{ref}}(p_i, p_j) = 1/N_s \sum_{k=1}^{N_s} C_k(\hat{g}_k(p_i), \hat{g}_k(p_j)), \quad p_i, p_j \in S_{\text{ref}} \tag{3}$$

In the M-step, the estimate $\hat{g}$ is refined to maximize the likelihood of the full data:

$$\hat{g} = \arg \max_{g=(g_1, \cdots, g_{N_s})} \log P(\overline{C}_{\text{ref}}, C_1, C_2, \cdots, C_{N_s}|g) \tag{4a}$$

$$= \arg \min_{g=(g_1, \cdots, g_{N_s})} \sum_{k=1}^{N_s} \sum_{p_i, p_j \in S_{\text{ref}}} (\overline{C}_{\text{ref}}(p_i, p_j) - C_k(g_k(p_i), g_k(p_j)))^2 \tag{4b}$$

where we have assumed that the noise in (1) is i.i.d. Gaussian. Because (4b) decouples, we can optimize over each subject's warp separately, i.e., these optimizations can be done in parallel:

$$\hat{g}_k = \arg \min_{g_k} \sum_{p_i, p_j \in S_{\text{ref}}} (\overline{C}_{\text{ref}}(p_i, p_j) - C_k(g_k(p_i), g_k(p_j)))^2 \tag{5}$$

However, an interesting alternative is to perform these sequentially with an E-step after each that updates the reference estimate $\overline{C}_{\text{ref}}$. This also allows some other interesting adaptations. We note:

$$(\overline{C}_{\text{ref}}(p_i, p_j) - C_k(g_k(p_i), g_k(p_j)))^2 \quad \propto \quad (\overline{C}_k(p_i, p_j) - C_k(g_k(p_i), g_k(p_j)))^2 \tag{6a}$$

where

$$\overline{C}_k(p_i, p_j) = \frac{1}{(N_s-1)} \sum_{n \neq k} C_n(\hat{g}_n(p_i), \hat{g}_n(p_j)), \quad p_i, p_j \in M_{\text{ref}}, \tag{7}$$

is the *leave-one-out template* for subject $k$, which is indepedendent of $g_k$. Thus, we replace (5) by:

$$\hat{g}_k = \arg \min_{g_k} \sum_{p_i, p_j \in S_{\text{ref}}} (\overline{C}_k(p_i, p_j) - C_k(g_k(p_i), g_k(p_j)))^2 \tag{8}$$

From (5) and (8) we observe that the multi-subject alignment problem reduces to a sequence of pairwise registrations, each of which registers one subject to an average of connectivity matrices. If we use (5), each round of pairwise registrations can be done in parallel and the results used to update the average template. The difficulty is the computational update of $\overline{C}_{\text{ref}}$. Alternatively, using (8) we do the pairwise registrations sequentially and compute a new leave-one-out template after each registration. This is the approach we pursue. An algorithm for solving the pairwise registration is derived in the next section and we examine the computation of leave-one-out templates in §4.

## 3 Pairwise Cortical Alignment

We now develop an algorithm for aligning one subject, with connectivity $C_F$, to a reference, with connectivity $C_R$, with $C_F, C_R \in \mathbf{R}^{N_v \times N_v}$. For concreteness, from this point forward we let $\sigma(f_1, f_2) = \text{corr}(f_1, f_2)$ and assume that the time series have zero mean and unit norm.

A function $g \colon M_R \to S_F$ maps a reference mesh point $p_i \in M_R$ to $g(p_i) \in S_F$. By interpolating the floating subject's times series at the points $g(p_i) \in S_F$ we obtain the associated warped functional connectivity: $\tilde{C}_F = [\sigma(f^F_{g(p_i)}, f^F_{g(p_j)})]$. We seek $\hat{g}$ that best matches $\tilde{C}_F$ to $C_R$ in the sense:

$$\hat{g} = \arg\min_g \|\tilde{C}_F - C_R\|^2_f + \lambda \mathrm{Reg}(g) \tag{9}$$

Here $\| \cdot \|_f$ is the matrix Frobenius norm and the regularization term $\mathrm{Reg}(g)$ serves as a prior over the space of allowable mappings. In the following steps, we examine how to efficiently solve (9).

**Step 1: Parameterizing the dependence of $\tilde{C}_F$ on the warp.** We first develop the dependence of the matrix $\tilde{C}_F$ on the warping function $g$. This requires specifying how the time series at the warped points $g(p_i) \in S_F$ is interpolated using the time series data $\{f^F_i \in \mathbb{R}^{N_t}, i = 1, \ldots, N_v\}$ at the mesh points $\{p^F_i \in M_F, i = 1, \ldots, N_v\}$. Here, we employ linear interpolation with a spherical kernel $\Phi$: $f^F(p) = \sum^{N_v}_{i=1} f^F_i \Phi(p, p_i), p \in S_F$. The kernel should be matched to the following specific objectives: (a) The kernel should be monomodal. Since the gradient of the registration objective depends on the derivative of the interpolation kernel, this will reduce the likelihood of the algorithm converging to a local minimum; (b) The support of the kernel should be finite. This will limit interpolation complexity. However, as the size of the support decreases, so will the capture range of the algorithm. At the initial stages of the algorithm, the kernel should have a broad extent, due to higher initial uncertainty, and become increasingly more localized as the algorithm converges. Thus, (c) The support of the kernel should be easily adjustable.

With these considerations in mind, we select $\Phi(p, p_i)$ to be a spherical radial basis function $\Phi_i :$ $S^2 \to \mathbb{R}$ centered at $p_i \in S^2$ and taking the form: $\Phi_i(p) = \varphi(d(p, p_i)), p \in S^2$, where $\varphi : [0, \pi] \to \mathbb{R}$ and $d(p, p_i)$ is the spherical geodesic distance between $p$ and $p_i$ [16]. Then $\Phi_i(p)$ is monomodal with a maximum at $p_i$, it depends only on the distance between $p$ and $p_i$ and is radially symmetric. In detail, we employ the particular spherical radial basis function:

$$\Phi_i(p) = \varphi(d(p, p_i)) = (1 - (2/r) \sin(d(p, p_i)/2))^4_+ ((8/r) \sin(d(p, p_i)/2) + 1) \tag{10}$$

where $r$ is a fixed parameter, and $(a)_+ = a\mathbf{1}\{a \geq 0\}$. $\Phi_i(p)$ has two continuous derivatives and its support is $\{p \in S^2 : d(p, p_i) < 2\sin^{-1}(r/2)\}$. Note that the support can be easily adjusted through the parameter $r$. So the kernel has all of our desired properties.

We can now make the dependence of $\tilde{C}_F$ on $g$ more explicit. Let $T_F = [f^F_1, f^F_2, \cdots, f^F_{N_v}]$. Then $\tilde{T}_F = \begin{bmatrix} f^F(g(p_1)) & f^F(g(p_2)) & \cdots & f^F(g(p_{N_v})) \end{bmatrix} = T_F A$ where $A = [\Phi_i(g(p_j))]$ is the $N_v \times N_v$ matrix of interpolation coefficients dependent on $g$ and the interpolation kernel. Next, noting that $C_F = T^T_F T_F$, we use $A$ to write the post-warp correlation matrix as:

$$\tilde{C}_F = DA^T C_F A D \tag{11}$$

where $D = \mathrm{diag}(d_1, d_2, \cdots, d_{N_v})$ serves to normalize the updated data to unit norm: $d_j = \|f^F(g(p_j))\|^{-1}$. Finally, we use $\tilde{A} = AD$ to write:

$$\|\tilde{C}_F - C_R\|^2_f = \|\tilde{A}^T C_F \tilde{A} - C_R\|^2_f \tag{12}$$

Here, (12) encodes the dependence of the registration objective on $g$ through the matrix $\tilde{A}$. It is also important to note that since the interpolation kernel is locally supported, $\tilde{A}$ is a sparse matrix.

**Step 2: Efficient Representation/Computation of the Registration Objective.** We now consider the $N_v \times N_v$ matrices $C_F$ and $C_R$. At a spatial resolution of 2 mm, the spherical model of human cortex can yield $N_v \approx 72,000$ total mesh points. In this situation, direct computation with $C_F$ and $C_R$ is prohibitive. Hence we need an efficient way to represent and compute the objective (12).

For fMRI data it is reasonable to assume that $N_t \ll N_v$. Hence, since the data has been centered, the rank of $C_F = T^T_F T_F$ and of $C_R = T^T_R T_R$ is at most $N_t - 1$. For simplicity, we make the reasonable assumption that $\mathrm{rank}(T_F) = \mathrm{rank}(T_R) = d$. Then $C_F$ and $C_R$ can be efficiently represented by compact $d$-dimensional SVDs $C_F = V_F \Sigma_F V^T_F$ and $C_R = V_R \Sigma_R V^T_R$. Moveover, these can be computed directly from SVDs of the data matrices: $T_F = U_{T_F} \Sigma_{T_F} V^T_{T_F}$ and $T_R = U_{T_R} \Sigma_{T_R} V^T_{T_R}$. In detail: $V_F = V_{T_F}, V_R = V_{T_R}, \Sigma_F = \Sigma^T_{T_F} \Sigma_{T_F}$, and $\Sigma_R = \Sigma^T_{T_R} \Sigma_{T_R}$.

The above representation avoids computing $C_F$ and $C_R$, but we must also show that it enables efficient evaluation of (12). To this end, introduce the following linear transformation:

$$B = W_F^T \tilde{A} W_R \qquad (13)$$

where $W_F = \begin{bmatrix} V_F & V_F^\perp \end{bmatrix}$, $W_R = \begin{bmatrix} V_R & V_R^\perp \end{bmatrix}$, are orthogonal with the $N_v - d$ columns of $V_F^\perp$ and $V_R^\perp$ forming orthonormal bases for $\mathrm{range}(V_F)^\perp$ and $\mathrm{range}(V_R)^\perp$, respectively. Write $B$ as:

$$B = \begin{bmatrix} B_1 & B_2 \\ B_3 & B_4 \end{bmatrix} \qquad (14)$$

with $B_1 \in \mathbb{R}^{d \times d}$, $B_2 \in \mathbb{R}^{d \times N_v}$, $B_3 \in \mathbb{R}^{(N_v - d) \times d}$ and $B_4 \in \mathbb{R}^{(N_v - d) \times (N_v - d)}$. Substituting (13) and (14) into (12) and simplifying yields:

$$\|\tilde{C}_F - C_R\|_f^2 = \|B_1^T \Sigma_F B_1 - \Sigma_R\|_f^2 + 2\|B_1^T \Sigma_F B_2\|_f^2 + \|B_2^T \Sigma_F B_2\|_f^2 \qquad (15)$$

with

$$B_1 = V_F^T \tilde{A} V_R \quad \text{and} \quad B_2 = V_F^T \tilde{A} V_R^\perp \qquad (16)$$

The $d \times d$ matrix $B_1$ is readily computed since $V_F$, $V_R$ are of manageable size. Computation of the $d \times N_v$ matrix $B_2$ depends on $V_R^\perp$. This has ON columns spanning the $N_v - d$ dimensional subspace $\mathrm{null}(C_R)$. Since there is residual freedom in the choice of $V_R^\perp$ and $B_2$ is large, its selection merits closer examination. Now (16) can be viewed as a projection of the rows of $V_F^T \tilde{A}$ onto the columns of $V_R$ and $V_R^\perp$. The columns of $\tilde{A} V_F - V_R B_1^T$ lie in $\mathrm{null}(C_R)$ and $B_2^T = (V_R^\perp)^T (\tilde{A} V_F - V_R B_1^T)$. Hence a $QR$-factorization $QR = \tilde{A}^T V_F - V_R B_1^T$ yields $d$ ON vectors in $\mathrm{null}(C_R)$. Choosing these as the first $d$ columns of $V_R^\perp$, yields $B_2 = [R\,,0]$, i.e., $B_2$ is very sparse.

In summary, we have derived the following efficient means of evaluating the objective. By one-time preprocessing of the time series data we obtain $\Sigma_F, \Sigma_R$ and $V_F, V_R$. Then given a warp $g$, we compute: the interpolation matrix $\tilde{A}$, $B_1 = V_F^T \tilde{A} V_R$, and finally $B_2$ via QR factorization of $\tilde{A}^T V_F - V_R B_1^T$. Then we evaluate (15).

**Step 3: The Transformation Space and Regularization.** We now examine the specification of $g$ in greater detail. We allow each mesh point to move freely (locally) in two directions. The use of such nonlinear warp models for inter-subject cortical alignment has been validated over, for example, rigid-body transformations [17]. To specify $g$, we first need to set up a coordinate system on the sphere. Let $U = \{(\phi, \theta); 0 < \phi < \pi, 0 < \theta < 2\pi\}$. Then the sphere can be parameterized by $x \colon U \to \mathbb{R}^3$ with $x(\phi, \theta) = (\sin\phi\cos\theta, \sin\phi\sin\theta, \cos\phi)$. Here, $\phi$ is a zenith angle measured against one of the principal axes, and $\theta$ is an azimuthal angle measured in one of the projective planes (i.e., $xy$-plane, $xz$-plane, or $yz$-plane). Note that $x$ omits a semicircle of $S^2$; so at least two such parameterizations are required to cover the entire sphere [18].

Consider $p_i \in S^2$ parameterized by $x(\phi, \theta)$ such that $p_i = x(\phi_i, \theta_i)$. Then the warp field at $p_i$ is:

$$g(p_i) = x(\phi_i + \Delta\phi_i, \theta_i + \Delta\theta_i) = x(\tilde{\phi}_i, \tilde{\theta}_i) \qquad (17)$$

for displacements $\Delta\phi_i$ and $\Delta\theta_i$. The warp $g$ is thus parameterized by: $\{\tilde{\phi}_i, \tilde{\theta}_i, i = 1, \dots, N_v\}$.

The warp $g$ must be regularized to avoid undesired topological distortions (e.g. folding and excessive expansion) and to avoid over-fitting the data. This is achieved by adding a regularization term to the objective that penalizes such distortions. There are several ways this can be done. Here we follow [14] and regularize $g$ by penalizing both metric and areal distortion. The metric distortion term penalizes warps that disrupt local distances between neighboring mesh nodes. This has the effect of limiting the expansion/contraction of cortex. The areal distortion term seeks to preserve a consistent orientation of the surface. Given a triangularization of the spherical mesh, each triangle is given an oriented normal vector that initially points radially outward from the sphere. Constraining the oriented area of all triangles to be positive prevents folds in the surface [14].

**Step 4: Optimization of the objective.** We optimize (3) over $g$ by gradient descent. Denote the objective by $S(g)$, let $\tilde{a}_{ij} = a_{ij} d_j$ be the $(i, j)$-th entry of $\tilde{A} = AD$ and $a(p) = [\Phi_1(p) \quad \Phi_2(p) \quad \cdots \quad \Phi_{N_v}(p)]^T$. From the parameterization of the warp (17), we see that $\tilde{a}_{ij} =$

| Algorithm 1 Pairwise algorithm | Algorithm 2 Multi-subject algorithm |
|---|---|
| 1: Given: SVD of floating dataset $\Sigma_F$, $V_F$ and reference dataset $\Sigma_R$, $V_R$<br>2: Given: Initial warp estimate $g^{(0)}$<br>3: Given: Sequence $r_1 > r_2 > \cdots > r_M$ of spatial resolutions<br>4: **for** $m = 1$ to $M$ **do**<br>5:    Set the kernel $\Phi_i$ in (10), with $r = r_m$<br>6:    Smooth the reference to resolution $r_m$<br>7:    Solve for $\hat{g}$ in (9) by gradient descent with initial condition $g^{(m-1)}$<br>8:    Set $g^{(m)} = \hat{g}$<br>9: **end for**<br>10: Output result: $g^{(M)}$ | 1: Given: SVD of datasets, $\{\Sigma_k, V_k\}_{k=1}^{N_s}$<br>2: Initialize $g_k^{(0)}$ to identity, $k = 1, \ldots, N_s$<br>3: **for** $t = 1$ to $T$ **do**<br>4:    **for** $k = 1$ to $N_s$ **do**<br>5:       Construct $\overline{C}_k$ as explained in §4<br>6:       Align $C_k$ to $\overline{C}_k$ by Algorithm 1 with initial condition $g_k^{(t-1)}$<br>7:       Set $g_k^{(t)}$ to the output of the alignment<br>8:       Use $g_k^{(t)}$ to update $\Sigma_k$, $V_k$<br>9:    **end for**<br>10: **end for**<br>11: Output result: $g = \{g_1^{(T)}, \ldots, g_{N_s}^{(T)})$ |

*Figure 1:* The registration algorithms.

$\Phi_i(x(\widetilde{\phi}_j, \widetilde{\theta}_j))\|T_F a(x(\widetilde{\phi}_j, \widetilde{\theta}_j))\|^{-1}$ depends only on the warp parameters of the $j^{th}$ mesh node, $\widetilde{\phi}_j$ and $\widetilde{\theta}_j$. Then, by the chain rule, the partial derivative of $S(g)$ with respect to $\widetilde{\phi}_j$ is given by:

$$\frac{\partial S(g)}{\partial \widetilde{\phi}_j} = \sum_{i=1}^{N_v} \frac{\partial \|\tilde{C}_F - C_R\|_f^2}{\partial \widetilde{a}_{ij}} \frac{\partial \widetilde{a}_{ij}}{\partial \widetilde{\phi}_j} + \lambda \frac{\partial \mathrm{Reg}(g)}{\partial \widetilde{\phi}_j} \tag{18}$$

A similar expression is obtained for the partial derivative with respect to $\widetilde{\theta}_j$. Since the interpolation kernel is supported locally, the summation in (18) is taken over a small number of terms. A full expression for $\partial S / \partial \widetilde{\phi}_j$ is given in the supplemental, and that of $\partial \mathrm{Reg}(g) / \partial \widetilde{\phi}_j$ in [14].

To help avoid local minima we take a multi-resolution optimization approach [19]. The registration is run on a sequence of spatial resolutions $r_1 > r_2 > \cdots > r_M$, with $r_M$ given by the original resolution of the data. The result at resolution $r_m$ is used to initialize the alignment at resolution $r_{m+1}$. The alignment for $r_m$ is performed by matching the kernel parameter $r$ in (10) to $r_m$. Note that the reference dataset is also spatially smoothed at each $r_m$ by the transformation in (11), with $A = [a(p_1)\, a(p_2)\, \cdots\, a(p_{N_v})]$. The pairwise algorithm is summarized as Algorithm 1 in Figure 1.

## 4  Multi-Subject Alignment: Computing Leave-one-out Templates

We now return to the multi-subject alignment problem, which is summarized as Algorithm 2 in Figure 1. It only remains to discuss efficient computation of the leave-one-out-template (7). Since $\overline{C}_k$ is an average of $N_s - 1$ positive semi-definite matrices each of rank $d$, the rank $\overline{d}$ of $\overline{C}_k$ is bounded as follows $d \leq \overline{d} \leq (N_s - 1)d$. Assume that $\widetilde{C}_n$, the connectivity matrix of subject $n$ after warp $g_n$ (see (11)), has an efficient $d \ll N_v$ dimensional SVD representation $\widetilde{C}_n = \widehat{V}_n \widetilde{\Sigma}_n \widetilde{V}_n^T$.

To compute the SVD for $\overline{C}_k$, we exploit the sequential nature of the multi-subject alignment algorithm by refining the SVD of the leave-one-out template for subject $k-1$, $\overline{C}_{k-1} = \overline{V}_{k-1}\overline{\Sigma}_{k-1}\overline{V}_{k-1}^T$, computed in the previous iteration. This is achieved by expressing $\overline{C}_k$ in terms of $\overline{C}_{k-1}$:

$$\overline{C}_k = \overline{C}_{k-1} + \tfrac{1}{N_s - 1}(\widetilde{C}_{k-1} - \widetilde{C}_k) \tag{19}$$

and computing matrix decompositions for the singular vectors of $\widetilde{C}_{k-1}$ and $\widetilde{C}_k$ in terms of $\overline{V}_{k-1}$:

$$\widetilde{V}_{k-1} = \overline{V}_{k-1}P_{k-1} + Q_{k-1}R_{k-1} \tag{20a}$$
$$\widetilde{V}_k = \overline{V}_{k-1}P_k \tag{20b}$$

where $P_j = \overline{V}_{k-1}^T \widetilde{V}_j \in \mathbb{R}^{\overline{d} \times d}$, for $j = k-1, k$, projects the columns of $\widetilde{V}_j$ onto the columns of $\overline{V}_{k-1}$. The second term of (20a), $Q_{k-1}R_{k-1}$, is the QR-decomposition of the residual components

of $\widetilde{V}_{k-1}$ after projection onto $\mathrm{range}(\overline{V}_{k-1})$. Since $\overline{C}_{k-1}$ is an average of positive semi-definite matrices that includes $\widetilde{C}_k$, we are sure that $\mathrm{range}(\widetilde{V}_k) \subseteq \mathrm{range}(\overline{V}_{k-1})$, (supplementary material).

Using the matrix decompositions (20a) and (20b), $\overline{C}_k$ in (19) above can be expressed as:

$$\overline{C}_k = \begin{bmatrix} \overline{V}_{k-1} & Q_{k-1} \end{bmatrix} G \begin{bmatrix} \overline{V}_{k-1} & Q_{k-1} \end{bmatrix}^T \tag{21}$$

where $G$ is the symmetric $(\overline{d} + d) \times (\overline{d} + d)$ matrix:

$$G = \begin{bmatrix} \overline{\Sigma}_{k-1} & 0 \\ 0 & 0 \end{bmatrix} + \frac{1}{N_s - 1} \left( \begin{bmatrix} P_{k-1}\widetilde{\Sigma}_{k-1}P_{k-1}^T & P_{k-1}\widetilde{\Sigma}_{k-1}R_{k-1}^T \\ R_{k-1}\widetilde{\Sigma}_{k-1}P_{k-1}^T & R_{k-1}\widetilde{\Sigma}_{k-1}R_{k-1}^T \end{bmatrix} - \begin{bmatrix} P_k\widetilde{\Sigma}_kP_k^T & 0 \\ 0 & 0 \end{bmatrix} \right) \tag{22}$$

We now compute the SVD of $G = V_G \Sigma_G V_G^T$. Then, using (21), we obtain the SVD for $\overline{C}_k$ as:

$$\overline{V}_k = \begin{bmatrix} \overline{V}_{k-1} & Q_{k-1} \end{bmatrix} V_G \quad \text{and} \quad \overline{\Sigma}_k = \Sigma_G \tag{23}$$

For a moderate number of subjects, $(\overline{d} + d) \leq N_s d \ll N_v$, this approach is more efficient than a brute-force $O(N_v^3)$ SVD. Additionally, it works directly on the singular values $\widetilde{\Sigma}_k$ and vectors $\widetilde{V}_k$ of each warped connectivity matrix $\widetilde{C}_k$, alleviating the need to store large $N_v \times N_v$ matrices.

## 5 Experimental Results

We tested the algorithm using fMRI data collected from 10 subjects viewing a movie split into 2 sessions separated by a short break. The data was preprocessed following [5]. For each subject, a structural scan was acquired before each session, from which the cortical surface model was derived (§2) and then anatomically aligned to a template using FreeSurfer (Fischl, http://surfer.nmr.mgh.harvard.edu). Similar to [5], we find that anatomical alignment based on cortical curvature serves as a superior starting point for functional alignment over Talairach alignment.

First, functional connectivity was found for each subject and session: $C_{k,i}$, $k = 1, \ldots, N_s$, $i = 1, 2$. These were then aligned within subjects, $C_{k,1} \leftrightarrow C_{k,2}$, and across subjects, $C_{k,1} \leftrightarrow C_{j,2}$, using Algorithm 1. Since the data starts in anatomical correspondence, we expect small warp displacements within subject and larger ones across subjects. The mean intra-subject warp displacement was $0.72$ mm ($\sigma = 0.48$), with $77\%$ of the mesh nodes warped less than 1 mm and fewer than $1.5\%$ warped by more than the data spatial resolution (2 mm). In contrast, the mean inter-subject warp displacement was $1.46$ mm ($\sigma = 0.92$ mm), with $22\%$ of nodes warped more than 2 mm. See Figures 2(a)-(b).

In a separate analysis, each subject was aligned to its leave-one-out template on each session using Algorithm 1, yielding a set of warps $g_{k,i}(p_j)$, $k = 1, \ldots, N_s$, $i = 1, 2$, $j = 1, \ldots, N_v$. To evaluate the consistency of the correspondence derived from different sessions, we compared the warps $g_{k,1}$ to $g_{k,2}$ for each subject $k$. Here, we only consider nodes that are warped by at least the data resolution. This analysis provides a measure of the sensitivity to noise present in the fMRI data. At node $p_j$, we compute the angle $0 \leq \theta \leq \pi$ between the warp tangent vectors of $g_{k,1}(p_j)$ and $g_{k,2}(p_j)$. This measures the consistency of the direction of the warp across sessions: smaller values of $\theta$ suggest a greater warp coherence across sessions. Figure 2(c) shows a histogram of $\theta$ averaged across the cortical nodes of all 10 subjects. The tight distribution centered near $\theta = 0$ suggests significant consistency in the warp direction across sessions. In particular, $93\%$ of the density for $\theta$ lies inside $\pi/2$, $81\%$ inside $\pi/4$, and $58\%$ inside $\pi/8$. As a secondary comparison, we compute a normalized consistency measure $\mathrm{WNC}(p_j) = d(g_{k,1}(p_j), g_{k,2}(p_j))/(d(g_{k,1}(p_j), p_j) + d(g_{k,2}(p_j), p_j))$, where $d(\cdot, \cdot)$ is spherical geodesic distance. The measure takes variability in both warp angle and magnitude into account; it is bounded between 0 and 1, and $\mathrm{WNC}(p_j) = 0$ only if $g_{k,1}(p_j) = g_{k,2}(p_j)$. A histogram for WNC is given in 2(d); WNC exhibits a peak at $0.15$, with a mean of $0.28$ ($\sigma = 0.22$).

Finally, Algorithm 2 was applied to the first session fMRI data to learn a set of warps $g = (g_1, \ldots, g_{N_s})$ for 10 subjects. The alignment required approximately 10 hours on a Intel 3.8GHz Nehalem quad-core processor with 12GB RAM. To evaluate the alignment, we apply the warps to the held out second session fMRI data, where subjects viewed a different segment of the movie. This warping yields data $\{f_{g_k(p_i)}^k\}$ for each subject $k$, with interpolation performed in the original volume to avoid artificial smoothing. The cross-validated inter-subject correlation $\mathrm{ISC}(p_i)$ is the mean

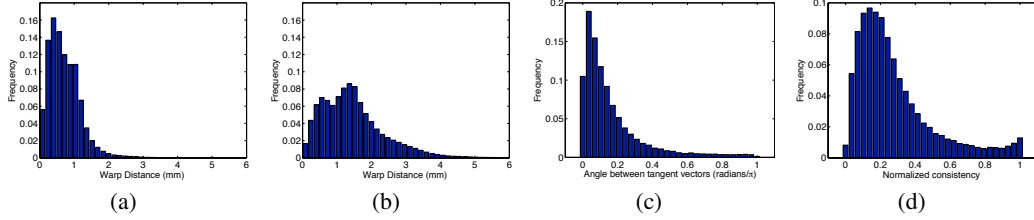

*Figure 2:* Consistency Histograms. (a) Intra-subject warp distances; (b) Inter-subject warp distances; (c) Angle between warp vectors across sessions; (d) Across-session normalized warp consistency measure WNC.

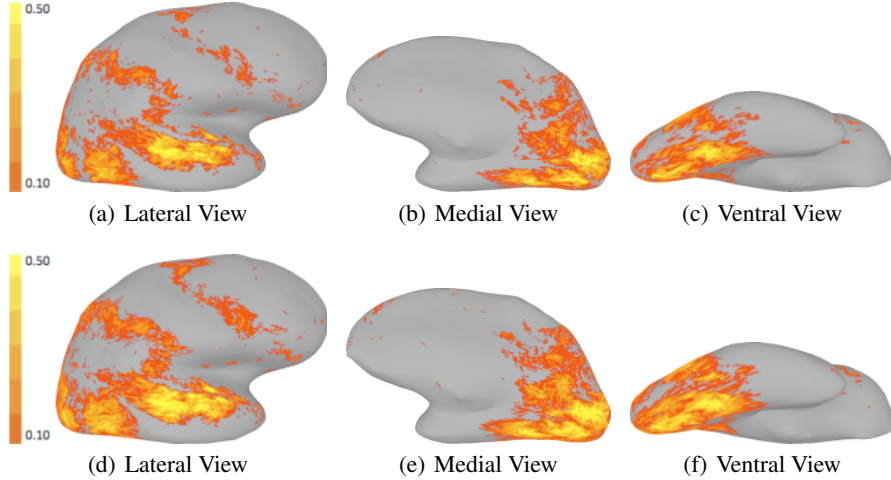

*Figure 3:* Map of ISC on right cortical hemisphere, alignment: anatomical (top), functional (bottom).

correlation of each subject's functional time series with the mean time series of the other subjects:

$$\mathrm{ISC}(p_i) = (1/N_s) \sum_{k=1}^{N_s} \mathrm{corr}(f_{g_k(p_i)}^{k}, \sum_{n \neq k} f_{g_n(p_i)}^{n}), \ p_i \in M_{\mathrm{ref}} \tag{24}$$

We also compute the mean inter-subject correlation, $\mathrm{ISC} = (1/N_v) \sum_{i=1}^{N_v} \mathrm{ISC}(p_i)$.

We compare the cross-validated ISC map with the ISC map of the second session movie viewing computed under anatomical correspondence. Mean ISC improved by $18\%$, from $0.072$ to $0.085$. In addition, the number of significant inter-subject correlations ($\mathrm{ISC}(p_i) > 0.1$, $P < 0.01$) increased by $22.9\%$, from $19,362$ to $23,789$. Figure 3 shows the ISC maps computed under anatomical alignment and functional alignment on the inflated right cortical hemisphere. As expected, the areas of improvement in inter-subject correlation are consistent with the extrinsic regions of cortex [6].

## 6 Conclusion

We have proposed a novel cortical registration algorithm that produces a functional correspondence across a set of subjects. The algorithm uses the fMRI data directly to align the spatial patterns of functional response elicited by a movie viewing. Despite the high-dimensionality of the data under consideration, the algorithm is efficient in both space and time complexity.

By comparing the inter-subject alignments derived from different fMRI experimental sessions, we show that the correspondence is consistent and robust to noise and variability in the fMRI temporal response. We also cross-validate the correspondence on independent test data that was not used to derive the alignment. On the test data, the algorithm produces a consistent increase in inter-subject correlation of fMRI time series, suggesting that functional alignment of extrinsic regions of cortex that are directly driven by the movie viewing experiment, such as visual and auditory areas, is improved considerably. Further testing is warranted to evaluate improvement in intrinsic areas of cortex whose response is not temporally synchronized with the experimental stimulus.

## Footnotes

*This work was funded by a grant from the National Institute of Mental Health (5R01MH075706-02)

# References

[1] J. Talairach and P. Tournoux. *Co-planar Stereotaxic Atlas of the Human Brain*. Thieme Publishing Group, 1988.

[2] B. Fischl, R.B.H. Tootell, and A.M. Dale. High-resolution intersubject averaging and a coordinate system for the cortical surface. *Human Brain Mapping*, 8:272–284, 1999.

[3] J.D.G. Watson, R. Myers, R.S.F. Frackowiak, J.V. Hajnal, R.P. Woods, J.C. Mazziotta, S. Shipp, and S. Zeki. Area v5 of the human brain: evidence from a combined study using positron emission tomography and magnetic resonance imaging. *Cerebral Cortex*, 3:79–94, 1993.

[4] J. Rademacher, V.S. Caviness, H. Steinmetz, and A.M. Galaburda. Topographical variation of the human primary cortices: implications for neuroimaging, brain mapping and neurobiology. *Cerebral Cortex*, 3:313–329, 1995.

[5] M.R. Sabuncu, B.D. Singer, B. Conroy, R.E. Bryan, P.J. Ramadge, and J.V. Haxby. Function-based inter-subject alignment of human cortical anatomy. *Cerebral Cortex Advance Access published on May 6, 2009, DOI 10.1093/cercor/bhp085*.

[6] U. Hasson, Y. Nir, G. Fuhrmann, and R. Malach. Intersubject synchronization of cortical activity during natural vision. *Science*, 303:1634–1640, 2004.

[7] Y. Golland, S. Bentin, H. Gelbard, Y. Benjamini, R. Heller, Y. Nir, U. Hasson, and R. Malach. Extrinsic and intrinsic systems in the posterior cortex of the human brain revealed during natural sensory stimulation. *Cerebral Cortex*, 17:766–777, 2007.

[8] M.E. Raichle, A.M. MacLeod, A.Z. Snyder, W.J. Powers, D.A. Gusnard, and G.L. Shulman. A default mode of brain function. *PNAS*, 98:676–682, 2001.

[9] K.J. Friston. Functional and effective connectivity in neuroimaging. *Human Brain Mapping*, 2:56–78, 1994.

[10] Michael D. Greicius, Ben Krasnow, Allan L. Reiss, and Vinod Menon. Functional connectivity in the resting brain: A network analysis of the default mode hypothesis. *PNAS*, 100:253–258, 2003.

[11] J.L. Vincent, A.Z. Snyder, M.D. Fox, B.J. Shannon, J.R. Andrews, M.E. Raichle, and R.L. Buckner. Coherent spontaneous activity identifies a hippocampal-parietal memory network. *J. Neurophysiol*, 96:3517–3531, 2006.

[12] D.C. Van Essen, H.A. Drury, J. Dickson, J. Harwell, D. Hanlon, and C.H. Anderson. An integrated software suite for surface-based analyses of cerebral cortex. *J. Am. Med. Inform. Assoc.*, 8:443–459, 2001.

[13] A.M. Dale, B. Fischl, and M.I. Sereno. Cortical surface-based analysis. i. segmentation and surface reconstruction. *NeuroImage*, 9:179–194, 1999.

[14] B. Fischl, M.I. Sereno, and A.M. Dale. Cortical surface-based analysis. ii. inflation, flattening, and a surface-based coordinate system. *NeuroImage*, 9:195–207, 1999.

[15] G.J. McLachlan and T. Krishnan. *The EM Algorithm and Extensions*. Wiley, 1997.

[16] G.E. Fasshauer and L.L. Schumaker. Scattered data fitting on the sphere. *Proceedings of the international conference on mathematical methods for curves and surfaces II*, pages 117–166, 1998.

[17] B.A. Ardekani, A.H. Bachman, S.C. Strother, Y. Fujibayashi, and Y. Yonekura. Impact of inter-subject image registration on group analysis of fmri data. *International Congress Series*, 1265:49–59, 2004.

[18] M. Do Carmo. *Differential Geometry of Curves and Surfaces*. Prentice Hall, 1976.

[19] R. Bajcsy and S. Kovacic. Multiresolution elastic matching. *Computer Vision, Graphics, and Image Processing*, 46:1–21, 1989.

